# Using Tarjan's Red Rule for Fast Dependency Tree Construction

**Dan Pelleg and Andrew Moore**
School of Computer Science
Carnegie-Mellon University
Pittsburgh, PA 15213 USA
`dpelleg@cs.cmu.edu, awm@cs.cmu.edu`

## Abstract

We focus on the problem of efficient learning of dependency trees. It is well-known that given the pairwise mutual information coefficients, a minimum-weight spanning tree algorithm solves this problem exactly and in polynomial time. However, for large data-sets it is the construction of the correlation matrix that dominates the running time. We have developed a new spanning-tree algorithm which is capable of exploiting partial knowledge about edge weights. The partial knowledge we maintain is a probabilistic confidence interval on the coefficients, which we derive by examining just a small sample of the data. The algorithm is able to flag the need to shrink an interval, which translates to inspection of more data for the particular attribute pair. Experimental results show running time that is near-constant in the number of records, without significant loss in accuracy of the generated trees. Interestingly, our spanning-tree algorithm is based solely on Tarjan's red-edge rule, which is generally considered a guaranteed recipe for bad performance.

## 1 Introduction

Bayes' nets are widely used for data modeling. However, the problem of constructing Bayes' nets from data remains a hard one, requiring search in a super-exponential space of possible graph structures. Despite recent advances [1], learning network structure from big data sets demands huge computational resources. We therefore turn to a simpler model, which is easier to compute while still being expressive enough to be useful. Namely, we look at dependency trees, which are belief networks that satisfy the additional constraint that each node has at most one parent. In this simple case it has been shown [2] that finding the tree that maximizes the data likelihood is equivalent to finding a minimum-weight spanning tree in the attribute graph, where edge weights are derived from the mutual information of the corresponding attribute pairs.

Dependency tress are interesting in their own right, but also as initializers for Bayes' Net search, as mixture components [3], or as components in classifiers [4]. It is our intent to eventually apply the technology introduced in this paper to the full problem of Bayes Net structure search.

Once the weight matrix is constructed, executing a minimum spanning tree (MST) algo-

rithm is fast. The time-consuming part is the population of the weight matrix, which takes time $O(Rn^2)$ for $R$ records and $n$ attributes. This becomes expensive when considering datasets with hundreds of thousands of records and hundreds of attributes.

To overcome this problem, we propose a new way of interleaving the spanning tree construction with the operations needed to compute the mutual information coefficients. We develop a new spanning-tree algorithm, based solely on Tarjan's [5] red-edge rule. This algorithm is capable of using partial knowledge about edge weights and of signaling the need for more accurate information regarding a particular edge. The partial information we maintain is in the form of probabilistic confidence intervals on the edge weights; an interval is derived by looking at a sub-sample of the data for a particular attribute pair. Whenever the algorithm signals that a currently-known interval is too wide, we inspect more data records in order to shrink it. Once the interval is small enough, we may be able to prove that the corresponding edge is *not* a part of the tree. Whenever such an edge can be eliminated without looking at the full data-set, the work associated with the remainder of the data is saved. This is where performance is potentially gained.

We have implemented the algorithm for numeric and categorical data and tested it on real and synthetic data-sets containing hundreds of attributes and millions of records. We show experimental results of up to $5,000$-fold speed improvements over the traditional algorithm. The resulting trees are, in most cases, of near-identical quality to the ones grown by the naive algorithm.

Use of probabilistic bounds to direct structure-search appears in [6] for classification and in [7] for model selection. In a sequence of papers, Domingos et al. have demonstrated the usefulness of this technique for decision trees [8], $K$-means clustering [9], and mixtures-of-Gaussians EM [10]. In the context of dependency trees, Meila [11] discusses the discrete case that frequently comes up in text-mining applications, where the attributes are sparse in the sense that only a small fraction of them is true for any record. In this case it is possible to exploit the sparseness and accelerate the Chow-Liu algorithm.

Throughout the paper we use the following notation. The number of data records is $R$, the number of attributes $n$. When $x$ is an attribute, $x_i$ is the value it takes for the $i$-th record. We denote by $\rho_{xy}$ the correlation coefficient between attributes $x$ and $y$, and omit the subscript when it is clear from the context.

## 2   A slow minimum-spanning tree algorithm

We begin by describing our MST algorithm[1]. Although in its given form it can be applied to any graph, it is asymptotically slower than established algorithms (as predicted in [5] for all algorithms in its class). We then proceed to describe its use in the case where some edge weights are known not exactly, but rather only to lie within a given interval. In Section 4 we will show how this property of the algorithm interacts with the data-scanning step to produce an efficient dependency-tree algorithm.

In the following discussion we assume we are given a complete graph with $n$ nodes, and the task is to find a tree connecting all of its nodes such that the total tree weight (defined to be the sum of the weights of its edges) is minimized. This problem has been extremely well studied and numerous efficient algorithms for it exist.

We start with a rule to eliminate edges from consideration for the output tree. Following [5], we state the so-called "red-edge" rule:

**Theorem 1:**   The heaviest edge in any cycle in the graph is not part of the minimum

```
1. T ← an arbitrary spanning set of n − 1 edges.
   L ← empty set.
2. While |L̄| > n − 1 do:
        Pick an arbitrary edge e ∈ L̄ \ T.
        Let e′ be the heaviest edge on the path in T between the
        endpoints of e.
        If e is heavier than e′:
            L ← L ∪ {e}
        otherwise:
            T ← T ∪ {e} \ {e′}
            L ← L ∪ {e′}
3. Output T.
```

Figure 1: The MIST algorithm. At each step of the iteration, $T$ contains the current "draft" tree. $L$ contains the set of edges that have been proven to *not* be in the MST and so $L̄$ contains the set of edges that still have some chance of being in the MST. $T$ never contains an edge in $L$.

spanning tree.

Traditionally, MST algorithms use this rule in conjunction with a greedy "blue-edge" rule, which chooses edges for inclusion in the tree. In contrast, we will repeatedly use the red-edge rule until all but $n − 1$ edges have been eliminated. The proof this results in a minimum-spanning tree follows from [5].

Let $E$ be the original set of edges. Denote by $L$ the set of edges that have already been eliminated, and let $L̄ = E \setminus L$. As a way to guide our search for edges to eliminate we maintain the following invariant:

**Invariant 2:** At any point there is a spanning tree $T$, which is composed of edges in $L̄$.

In each step, we arbitrarily choose some edge $e$ in $L̄ \setminus T$ and try to eliminate it using the red-edge rule. Let $P$ be the path in $T$ between $e$'s endpoints. The cycle we will apply the red-edge rule to will be composed of $e$ and $P$. It is clear we only need to compare $e$ with the heaviest edge in $P$. If $e$ is heavier, we can eliminate it by the red-edge rule. However, if it is lighter, then we can eliminate the tree edge by the same rule. We do so and add $e$ to the tree to preserve Invariant 2. The algorithm, which we call Minimum Incremental Spanning Tree (MIST), is listed in Figure 1.

The MIST algorithm can be applied directly to a graph where the edge weights are known exactly. And like many other MST algorithms, it can also be used in the case where just the relative order of the edge weights is given. Now imagine a different setup, where edge weights are not given, and instead an oracle exists, who knows the exact values of the edge weights. When asked about the relative order of two edges, it may either respond with the correct answer, or it may give an inconclusive answer. Furthermore, a constant fee is charged for each query. In this setup, MIST is still suited for finding a spanning tree while minimizing the number of queries issued. In step 2, we go to the oracle to determine the order. If the answer is conclusive, the algorithm proceeds as described. Otherwise, it just ignores the "if" clause altogether and iterates (possibly with a different edge $e$).

For the moment, this setup may seem contrived, but in Section 4, we go back to the MIST algorithm and put it in a context very similar to the one described here.

## 3  Probabilistic bounds on mutual information

We now concentrate once again on the specific problem of determining the mutual information between a pair of attributes. We show how to compute it given the complete data, and how to derive probabilistic confidence intervals for it, given just a sample of the data.

As shown in [12], the mutual information for two jointly Gaussian numeric attributes $X$ and $Y$ is:

$$I(X;Y) = -\frac{1}{2}\ln(1 - \rho^2)$$

where the correlation coefficient $\rho = \rho_{XY} = \frac{\sum_{i=1}^{R}((x_i - \bar{x})(y_i - \bar{y}))}{\hat{\sigma}_X^2 \hat{\sigma}_Y^2}$ with $\bar{x}, \bar{y}, \hat{\sigma}_X^2$ and $\hat{\sigma}_Y^2$ being the sample means and variances for attributes $X$ and $Y$.

Since the $\log$ function is monotonic, $I(X;Y)$ must be monotonic in $|\rho|$. This is a sufficient condition for the use of $|\rho|$ as the edge weight in a MST algorithm. Consequently, the sample correlation can be used in a straightforward manner when the complete data is available. Now consider the case where just a sample of the data has been observed.

Let $x$ and $y$ be two data attributes. We are trying to estimate $\sum_{i=1}^{R} x_i \cdot y_i$ given the partial sum $\sum_{i=1}^{r} x_i \cdot y_i$ for some $r < R$. To derive a confidence interval, we use the Central Limit Theorem [2]. It states that given samples of the random variable $Z$ (where for our purposes $Z_i = x_i \cdot y_i$), the sum $\sum_i Z_i$ can be approximated by a Normal distribution with mean and variance closely related to the distribution mean and variance. Furthermore, for large samples, the sample mean and variance can be substituted for the unknown distribution parameters. Note in particular that the central limit theorem *does not require us to make any assumption about the Gaussianity of* $Z$. We thus can derive a two-sided confidence interval for $\sum_i Z_i = \sum_i x_i \cdot y_i$ with probability $1 - \delta$ for some user-specified $\delta$, typically 1%. Given this interval, computing an interval for $\rho$ is straightforward. Categorical data can be treated similarly; for lack of space we refer the reader to [13] for the details.

## 4  The full algorithm

As we argued, the MIST algorithm is capable of using partial information about edge weights. We have also shown how to derive confidence intervals on edge weights. We now combine the two and give an efficient dependency-tree algorithm.

We largely follow the MIST algorithm as listed in Figure 1. We initialize the tree $T$ in the following heuristic way: first we take a small sub-sample of the data, and derive point estimates for the edge weights from it. Then we feed the point estimates to a MST algorithm and obtain a tree $T$.

When we come to compare edge weights, we generally need to deal with two intervals. If they do not intersect, then the points in one of them are all smaller in value than any point in the other, in which case we can determine which represents a heavier edge. We apply this logic to all comparisons, where the goal is to determine the heaviest path edge $e'$ and to compare it to the candidate $e$. If we are lucky enough that all of these comparisons are conclusive, the amount of work we save is related to how much data was used in computing the confidence intervals — the rest of the data for the attribute-pair that is represented by the eliminated edge can be ignored.

However, there is no guarantee that the intervals are separated and allow us to draw meaningful conclusions. If they do not, then we have a situation similar to the inconclusive

oracle answers in Section 2. The price we need to pay here is looking at more data to shrink the confidence intervals. We do this by choosing one edge — either a tree-path edge or the candidate edge — for "promotion", and doubling the sample size used to compute the sufficient statistics for it. After doing so we try to eliminate again (since we can do this at no additional cost). If we fail to eliminate we iterate, possibly choosing a different candidate edge (and the corresponding tree path) this time. The choice of which edge to promote is heuristic, and depends on the expected success of resolution once the interval has shrunk. The details of these heuristics are omitted due to space constraints.

Another heuristic we employ goes as follows. Consider the comparison of the path-heaviest edge to an estimate of a candidate edge. The candidate edge's confidence interval may be very small, and yet still intersect the interval that is the heavy edge's weight (this would happen if, for example, both attribute-pairs have the same distribution). We may be able to reduce the amount of work by pretending the interval is narrower than it really is. We therefore trim the interval by a constant, parameterized by the user as $\epsilon$, before performing the comparison. This use of $\delta$ and $\epsilon$ is analogous to their use in "Probably Approximately Correct" analysis: on each decision, with high probability $(1 - \delta)$ we will make at worst a small mistake $(\epsilon)$.

## 5   Experimental results

In the following description of experiments, we vary different parameters for the data and the algorithm. Unless otherwise specified, these are the default values for the parameters. We set $\delta$ to $1\%$ and $\epsilon$ to $0.05$ (on either side of the interval, totaling $0.1$). The initial sample size is fifty records. There are $100,000$ records and $100$ attributes. The data is numeric. The data-generation process first generates a random tree, then draws points for each node from a normal distribution with the node's parent's value as the mean. In addition, any data value is set to random noise with probability $0.15$.

To construct the correlation matrix from the full data, each of the $R$ records needs to be considered for each of the $\binom{n}{2}$ attribute pairs. We evaluate the performance of our algorithm by adding the number of records that were actually scanned for all the attribute-pairs, and dividing the total by $R\binom{n}{2}$. We call this number the "data usage" of our algorithm. The closer it is to zero, the more efficient our sampling is, while a value of one means the same amount of work as for the full-data algorithm.

We first demonstrate the speed of our algorithm as compared with the full $O(Rn^2)$ scan. Figure 2 shows that the amount of data the algorithm examines is a constant that does not depend on the size of the data-set. This translates to relative run-times of $0.7\%$ (for the $37,500$-record set) to $0.02\%$ (for the $1,200,000$-record set) as compared with the full-data algorithm. The latter number translates to a $5,000$-fold speedup. Note that the reported usage is an average over the number of attributes. However this does not mean that the same amount of data was inspected for every attribute-pair — the algorithm determines how much effort to invest in each edge separately. We return to this point below.

The running time is plotted against the number of data attributes in Figure 3. A linear relation is clearly seen, meaning that (at least for this particular data-generation scheme) the algorithm is successful in doing work that is proportional to the number of tree edges.

Clearly speed has to be traded off. For our algorithm the risk is making the wrong decision about which edges to include in the resulting tree. For many applications this is an acceptable risk. However, there might be a simpler way to grow estimate-based dependency trees, one that does not involve complex red-edge rules. In particular, we can just run the original algorithm on a small sample of the data, and use the generated tree. It would certainly be fast, and the only question is how well it performs.

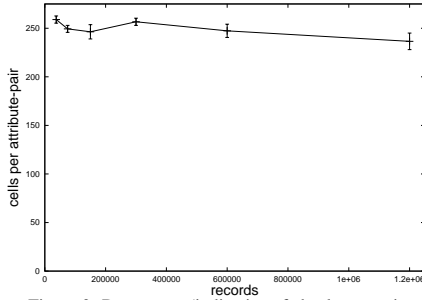

Figure 2: Data usage (indicative of absolute running time), in attribute-pair units per attribute.

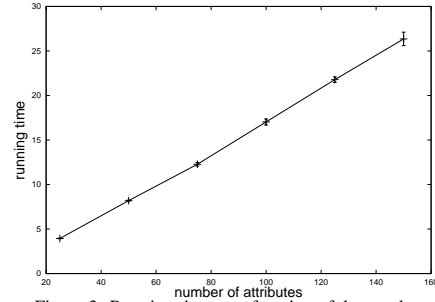

Figure 3: Running time as a function of the number of attributes.

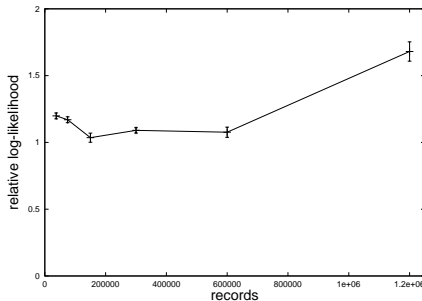

Figure 4: Relative log-likelihood vs. the sample-based algorithm. The log-likelihood difference is divided by the number of records.

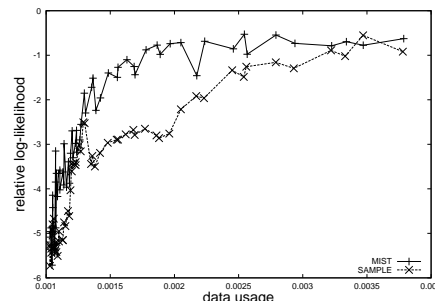

Figure 5: Relative log-likelihood vs. the sample-based algorithm, drawn against the fraction of data scanned.

To examine this effect we have generated data as above, then ran a 30-fold cross-validation test for the trees our algorithm generated. We also ran a sample-based algorithm on each of the folds. This variant behaves just like the full-data algorithm, but instead examines just the fraction of it that adds up to the total amount of data used by our algorithm. Results for multiple data-sets are in Figure 4. We see that our algorithm outperforms the sample-based algorithm, even though they are both using the same total amount of data. The reason is that using the same amount of data for all edges assumes all attribute-pairs have the same variance. This is in contrast to our algorithm, which determines the amount of data for each edge independently. Apparently for some edges this decision is very easy, requiring just a small sample. These "savings" can be used to look at more data for high-variance edges. The sample-based algorithm would not put more effort into those high-variance edges, eventually making the wrong decision. In Figure 5 we show the log-likelihood difference for a particular (randomly generated) set. Here, multiple runs with different $\delta$ and $\epsilon$ values were performed, and the result is plotted against the fraction of data used. The baseline (0) is the log-likelihood of the tree grown by the original algorithm using the full data. Again we see that MIST is better over a wide range of data utilization ratio.

Keep in mind that the sample-based algorithm has been given an unfair advantage, compared with MIST: it knows how much data it needs to look at. This parameter is implicitly passed to it from our algorithm, and represents an important piece of information about the data. Without it, there would need to be a preliminary stage to determine the sample size. The alternative is to use a fixed amount (specified either as a fraction or as an absolute count), which is likely to be too much or too little.

To test our algorithm on real-life data, we used various data-sets from [14, 15], as well as analyzed data derived from astronomical observations taken in the Sloan Digital Sky Survey. On each data-set we ran a 30-fold cross-validation test as described above. For

Table 1: Results, relative to the sample-based algorithm, on real data. "Type" means numerical or categorical data.

| NAME | ATTR. | RECORDS | TYPE | DATA USAGE | MIST BETTER? | SAMPLE BETTER? |
|---|---|---|---|---|---|---|
| CENSUS-HOUSE | 129 | 22784 | N | 1.0% | × | √ |
| COLORHISTOGRAM | 32 | 68040 | N | 0.5% | √ | × |
| COOCTEXTURE | 16 | 68040 | N | 4.6% | × | √ |
| ABALONE | 8 | 4177 | N | 21.0% | × | × |
| COLORMOMENTS | 10 | 68040 | N | 0.6% | × | √ |
| CENSUS-INCOME | 678 | 99762 | C | 0.05% | √ | × |
| COIL2000 | 624 | 5822 | C | 0.9% | √ | × |
| IPUMS | 439 | 88443 | C | 0.06% | √ | × |
| KDDCUP99 | 214 | 303039 | C | 0.02% | √ | × |
| LETTER | 16 | 20000 | N | 1.5% | √ | × |
| COVTYPE | 151 | 581012 | C | 0.009% | × | √ |
| PHOTOZ | 23 | 2381112 | N | 0.008% | √ | × |

each training fold, we ran our algorithm, followed by a sample-based algorithm that uses as much data as our algorithm did. Then the log-likelihoods of both trees were computed for the test fold. Table 1 shows whether the 99% confidence interval for the log-likelihood difference indicates that either of the algorithms outperforms the other. In seven cases the MIST-based algorithm was better, while the sample-based version won in four, and there was one tie. Remember that the sample-based algorithm takes advantage of the "data usage" quantity computed by our algorithm. Without it, it would be weaker or slower, depending on how conservative the sample size was.

## 6   Conclusion and future work

We have presented an algorithm that applies a "probably approximately correct" approach to dependency-tree construction for numeric and categorical data. Experiments in sets with up to millions of records and hundreds of attributes show it is capable of processing massive data-sets in time that is constant in the number of records, with just a minor loss in output quality.

Future work includes embedding our algorithm in a framework for fast Bayes' Net structure search.

A additional issue we would like to tackle is disk access. One advantage the full-data algorithm has is that it is easily executed with a single sequential scan of the data file. We will explore the ways in which this behavior can be attained or approximated by our algorithm.

While we have derived formulas for both numeric and categorical data, we currently do not allow both types of attributes to be present in a single network.

**Acknowledgments**

We would like to thank Mihai Budiu, Scott Davies, Danny Sleator and Larry Wasserman for helpful discussions, and Andy Connolly for providing access to data.

## Footnotes

[1]To be precise, we will use it as a *maximum* spanning tree algorithm. The two are interchangeable, requiring just a reversal of the edge weight comparison operator.

[2]One can use the weaker Hoeffding bound instead, and our implementation supports it as well, although it is generally much less powerful.

# References

[1] Nir Friedman, Iftach Nachman, and Dana Peér. Learning bayesian network structure from massive datasets: The "sparse candidate" algorithm. In *Proceedings of the 15th Conference on Uncertainty in Artificial Intelligence (UAI-99)*, pages 206–215, Stockholm, Sweden, 1999.

[2] C. K. Chow and C. N. Liu. Approximating discrete probability distributions with dependence trees. *IEEE Transactions on Information Theory*, 14:462–467, 1968.

[3] Marina Meila. *Learning with Mixtures of Trees*. PhD thesis, Massachusetts Institute of Technology, 1999.

[4] N. Friedman, M. Goldszmidt, and T. J. Lee. Bayesian Network Classification with Continuous Attributes: Getting the Best of Both Discretization and Parametric Fitting. In Jude Shavlik, editor, *International Conference on Machine Learning*, 1998.

[5] Robert Endre Tarjan. *Data structures and network algorithms*, volume 44 of *CBMS-NSF Reg. Conf. Ser. Appl. Math.* SIAM, 1983.

[6] Oded Maron and Andrew W. Moore. Hoeffding races: Accelerating model selection search for classification and function approximation. In Jack D. Cowan, Gerald Tesauro, and Joshua Alspector, editors, *Advances in Neural Information Processing Systems*, volume 6, pages 59–66, Denver, Colorado, 1994. Morgan Kaufmann.

[7] Andrew W. Moore and Mary S. Lee. Efficient algorithms for minimizing cross validation error. In *Proceedings of the 11th International Conference on Machine Learning (ICML-94)*, pages 190–198. Morgan Kaufmann, 1994.

[8] Pedro Domingos and Geoff Hulten. Mining high-speed data streams. In Raghu Ramakrishnan, Sal Stolfo, Roberto Bayardo, and Ismail Parsa, editors, *Proceedings of the 6th ACM SIGKDD International Conference on Knowledge Discovery and Data Mining (KDD-00)*, pages 71–80, N. Y., August 20–23 2000. ACM Press.

[9] Pedro Domingos and Geoff Hulten. A general method for scaling up machine learning algorithms and its application to clustering. In Carla Brodley and Andrea Danyluk, editors, *Proceeding of the 17th International Conference on Machine Learning*, San Francisco, CA, 2001. Morgan Kaufmann.

[10] Pedro Domingos and Geoff Hulten. Learning from infinite data in finite time. In *Proceedings of the 14th Neural Information Processing Systems (NIPS-2001)*, Vancouver, British Columbia, Canada, 2001.

[11] Marina Meila. An accelerated Chow and Liu algorithm: fitting tree distributions to high dimensional sparse data. In *Proceedings of the Sixteenth International Conference on Machine Learning (ICML-99)*, Bled, Slovenia, 1999.

[12] Fazlollah Reza. *An Introduction to Information Theory*, pages 282–283. Dover Publications, New York, 1994.

[13] Dan Pelleg and Andrew Moore. Using Tarjan's red rule for fast dependency tree construction. Technical Report CMU-CS-02-116, Carnegie-Mellon University, 2002.

[14] C.L. Blake and C.J. Merz. UCI repository of machine learning databases, 1998. http://www.ics.uci.edu/~mlearn/MLRepository.html.

[15] S. Hettich and S. D. Bay. The UCI KDD archive, 1999. http://kdd.ics.uci.edu.
